# How SVMs can estimate quantiles and the median

**Ingo Steinwart**
Information Sciences Group CCS-3
Los Alamos National Laboratory
Los Alamos, NM 87545, USA
ingo@lanl.gov

**Andreas Christmann**
Department of Mathematics
Vrije Universiteit Brussel
B-1050 Brussels, Belgium
andreas.christmann@vub.ac.be

## Abstract

We investigate quantile regression based on the pinball loss and the $\epsilon$-insensitive loss. For the pinball loss a condition on the data-generating distribution $P$ is given that ensures that the conditional quantiles are approximated with respect to $\|\cdot\|_1$. This result is then used to derive an oracle inequality for an SVM based on the pinball loss. Moreover, we show that SVMs based on the $\epsilon$-insensitive loss estimate the conditional median only under certain conditions on $P$.

## 1 Introduction

Let P be a distribution on $X \times Y$, where $X$ is an arbitrary set and $Y \subset \mathbb{R}$ is closed. The goal of quantile regression is to estimate the conditional quantile, *i.e.*, the set valued function

$$F_{\tau,\mathrm{P}}^*(x) := \left\{ t \in \mathbb{R} : \mathrm{P}\big((-\infty, t] \,|\, x\big) \geq \tau \text{ and } \mathrm{P}\big([t, \infty) \,|\, x\big) \geq 1 - \tau \right\}, \quad x \in X,$$

where $\tau \in (0, 1)$ is a fixed constant and $\mathrm{P}(\cdot \,|\, x)$, $x \in X$, is the (regular) conditional probability. For conceptual simplicity (though mathematically this is not necessary) we assume throughout this paper that $F_{\tau,\mathrm{P}}^*(x)$ consists of singletons, i.e., there exists a function $f_{\tau,\mathrm{P}}^* : X \to \mathbb{R}$, called the conditional $\tau$-quantile function, such that $F_{\tau,\mathrm{P}}^*(x) = \{f_{\tau,\mathrm{P}}^*(x)\}$, $x \in X$. Let us now consider the so-called $\tau$-*pinball loss* $L_\tau : \mathbb{R} \times \mathbb{R} \to [0, \infty)$ defined by $L_\tau(y, t) := \psi_\tau(y - t)$, where $\psi_\tau(r) = (\tau - 1)r$, if $r < 0$, and $\psi_\tau(r) = \tau r$, if $r \geq 0$. Moreover, given a (measurable) function $f : X \to \mathbb{R}$ we define the $L_\tau$-risk of $f$ by $\mathcal{R}_{L_\tau,\mathrm{P}}(f) := \mathbb{E}_{(x,y) \sim \mathrm{P}} L_\tau(y, f(x))$. Now recall that $f_{\tau,\mathrm{P}}^*$ is up to zero sets the *only* function that minimizes the $L_\tau$-risk, i.e. $\mathcal{R}_{L_\tau,\mathrm{P}}(f_{\tau,\mathrm{P}}^*) = \inf \mathcal{R}_{L_\tau,\mathrm{P}}(f) =: \mathcal{R}_{L_\tau,\mathrm{P}}^*$, where the infimum is taken over all $f : X \to \mathbb{R}$. Based on this observation several estimators minimizing a (modified) empirical $L_\tau$-risk were proposed (see [5] for a survey on both parametric and non-parametric methods) for situations where P is unknown, but i.i.d. samples $D := ((x_1, y_1), \ldots, (x_n, y_n))$ drawn from P are given. In particular, [6, 4, 10] proposed an SVM that finds a solution $f_{D,\lambda} \in H$ of

$$\arg\min_{f \in H} \lambda \|f\|_H^2 + \frac{1}{n} \sum_{i=1}^n L_\tau(y_i, f(x_i)), \tag{1}$$

where $\lambda > 0$ is a regularization parameter and $H$ is a reproducing kernel Hilbert space (RKHS) over $X$. Note that this optimization problem can be solved by considering the dual problem [4, 10], but since this technique is nowadays standard in machine learning we omit the details. Moreover, [10] contains an exhaustive empirical study as well some theoretical considerations.

Empirical methods estimating quantiles with the help of the pinball loss typically obtain functions $f_D$ for which $\mathcal{R}_{L_\tau,\mathrm{P}}(f_D)$ is close to $\mathcal{R}_{L_\tau,\mathrm{P}}^*$ with high probability. However, in general this only implies that $f_D$ is close to $f_{\tau,\mathrm{P}}^*$ in a very weak sense (see [7, Remark 3.18]), and hence there is so far only little justification for using $f_D$ as an estimate of the quantile function. Our goal is to address this issue by showing that under certain realistic assumptions on P we have an inequality of the form

$$\|f - f_{\tau,\mathrm{P}}^*\|_{L_1(\mathrm{P}_X)} \leq c_\mathrm{P} \sqrt{\mathcal{R}_{L_\tau,\mathrm{P}}(f) - \mathcal{R}_{L_\tau,\mathrm{P}}^*}. \tag{2}$$

We then use this inequality to establish an oracle inequality for SVMs defined by (1). In addition, we illustrate how this oracle inequality can be used to obtain learning rates and to justify a data-dependent method for finding the hyper-parameter $\lambda$ and $H$. Finally, we generalize the methods for establishing (2) to investigate the role of $\epsilon$ in the $\epsilon$-insensitive loss used in standard SVM regression.

## 2 Main results

In the following $X$ is an arbitrary, non-empty set equipped with a $\sigma$-algebra, and $Y \subset \mathbb{R}$ is a closed non-empty set. Given a distribution P on $X \times Y$ we further assume throughout this paper that the $\sigma$-algebra on $X$ is complete with respect to the marginal distribution $\mathrm{P}_X$ of P, i.e., every subset of a $\mathrm{P}_X$-zero set is contained in the $\sigma$-algebra. Since the latter can always be ensured by increasing the original $\sigma$-algebra in a suitable manner we note that this is not a restriction at all.

**Definition 2.1** *A distribution* Q *on* $\mathbb{R}$ *is said to have a* $\tau$-*quantile of type* $\alpha > 0$ *if there exists a* $\tau$-*quantile* $t^* \in \mathbb{R}$ *and a constant* $c_Q > 0$ *such that for all* $s \in [0, \alpha]$ *we have*

$$\mathrm{Q}\big((t^*, t^* + s)\big) \geq c_Q \, s \qquad and \qquad \mathrm{Q}\big((t^* - s, t^*)\big) \geq c_Q \, s. \tag{3}$$

It is not difficult to see that a distribution Q having a $\tau$-quantile of some type $\alpha$ has a unique $\tau$-quantile $t^*$. Moreover, if Q has a Lebesgue density $h_Q$ then Q has a $\tau$-quantile of type $\alpha$ if $h_Q$ is bounded away from zero on $[t^* - \alpha, t^* + \alpha]$ since we can use $c_Q := \inf\{h_Q(t) : t \in [t^* - \alpha, t^* + \alpha]\}$ in (3). This assumption is general enough to cover many distributions used in parametric statistics such as Gaussian, Student's $t$, and logistic distributions (with $Y = \mathbb{R}$), Gamma and log-normal distributions (with $Y = [0, \infty)$), and uniform and Beta distributions (with $Y = [0, 1]$).

The following definition describes distributions on $X \times Y$ whose conditional distributions $\mathrm{P}(\cdot \,|x)$, $x \in X$, have the same $\tau$-quantile type $\alpha$.

**Definition 2.2** *Let* $p \in (0, \infty]$, $\tau \in (0, 1)$, *and* $\alpha > 0$. *A distribution* P *on* $X \times Y$ *is said to have a* $\tau$-*quantile of* $p$-*average type* $\alpha$, *if* $\mathrm{Q}_x := \mathrm{P}(\cdot\,|x)$ *has* $\mathrm{P}_X$-*almost surely a* $\tau$-*quantile type* $\alpha$ *and* $b : X \to (0, \infty)$ *defined by* $b(x) := c_{\mathrm{P}(\cdot\,|x)}$, *where* $c_{\mathrm{P}(\cdot\,|x)}$ *is the constant in (3), satisfies* $b^{-1} \in L_p(\mathrm{P}_X)$.

Let us now give some examples for distributions having $\tau$-quantiles of $p$-average type $\alpha$.

**Example 2.3** *Let* P *be a distribution on* $X \times \mathbb{R}$ *with marginal distribution* $\mathrm{P}_X$ *and regular conditional probability* $\mathrm{Q}_x\big((-\infty, y]\big) := 1/(1 + e^{-z})$, $y \in \mathbb{R}$, *where* $z := \big(y - m(x)\big)/\sigma(x)$, $m : X \to \mathbb{R}$ *describes a location shift, and* $\sigma : X \to [\beta, 1/\beta]$ *describes a scale modification for some constant* $\beta \in (0, 1]$. *Let us further assume that the functions* $m$ *and* $\sigma$ *are measurable. Thus* $\mathrm{Q}_x$ *is a logistic distribution having the positive and bounded Lebesgue density* $h_{\mathrm{Q}_x}(y) = e^{-z}/(1 + e^{-z})^2$, $y \in \mathbb{R}$. *The* $\tau$-*quantile function is* $t^*(x) := f^*_{\tau, \mathrm{Q}_x} = m(x) + \sigma(x)\log(\frac{\tau}{1-\tau})$, $x \in X$, *and we can choose* $b(x) = \inf\{h_{\mathrm{Q}_x}(t) : t \in [t^*(x) - \alpha, t^*(x) + \alpha]\}$. *Note that* $h_{\mathrm{Q}_x}(m(x) + y) = h_{\mathrm{Q}_x}(m(x) - y)$ *for all* $y \in \mathbb{R}$, *and* $h_{\mathrm{Q}_x}(y)$ *is strictly decreasing for* $y \in [m(x), \infty)$. *Some calculations show*

$$b(x) = \min\big\{h_{\mathrm{Q}_x}(t^*(x) - \alpha), h_{\mathrm{Q}_x}(t^*(x) + \alpha)\big\} = \min\Big\{\frac{u_1(x)}{(1 + u_1(x))^2}, \frac{u_2(x)}{(1 + u_2(x))^2}\Big\} \in \Big(c_{\alpha, \beta}, \frac{1}{4}\Big),$$

*where* $u_1(x) := \frac{1-\tau}{\tau}e^{-\alpha/\sigma(x)}$, $u_2(x) := \frac{1-\tau}{\tau}e^{\alpha/\sigma(x)}$ *and* $c_{\alpha, \beta} > 0$ *can be chosen independent of* $x$, *because* $\sigma(x) \in [\beta, 1/\beta]$. *Hence* $b^{-1} \in L_\infty(\mathrm{P}_X)$ *and* P *has a* $\tau$-*quantile of* $\infty$-*average type* $\alpha$.

**Example 2.4** *Let* $\tilde{\mathrm{P}}$ *be a distribution on* $X \times Y$ *with marginal distribution* $\tilde{\mathrm{P}}_X$ *and regular conditional probability* $\tilde{\mathrm{Q}}_x := \tilde{\mathrm{P}}(\cdot\,|x)$ *on* $Y$. *Furthermore, assume that* $\tilde{\mathrm{Q}}_x$ *is* $\tilde{\mathrm{P}}_X$-*almost surely of* $\tau$-*quantile type* $\alpha$. *Let us now consider the family of distributions* P *with marginal distribution* $\tilde{\mathrm{P}}_X$ *and regular conditional distributions* $\mathrm{Q}_x := \tilde{\mathrm{P}}\big((\cdot - m(x))/\sigma(x)\,\big|\,x\big)$, $x \in X$, *where* $m : X \to \mathbb{R}$ *and* $\sigma : X \to (\beta, 1/\beta)$ *are as in the previous example. Then* $\mathrm{Q}_x$ *has a* $\tau$-*quantile* $f^*_{\tau, \mathrm{Q}_x} = m(x) + \sigma(x)f^*_{\tau, \tilde{\mathrm{Q}}_x}$ *of type* $\alpha\beta$, *because we obtain for* $s \in [0, \alpha\beta]$ *the inequality*

$$\mathrm{Q}_x\big((f^*_{\tau, \mathrm{Q}_x}, f^*_{\tau, \mathrm{Q}_x} + s)\big) = \tilde{\mathrm{Q}}_x\big((f^*_{\tau, \tilde{\mathrm{Q}}_x}, f^*_{\tau, \tilde{\mathrm{Q}}_x} + s/\sigma(x))\big) \geq b(x)s/\sigma(x) \geq b(x)\beta s.$$

*Consequently,* P *has a* $\tau$-*quantile of* $p$-*average type* $\alpha\beta$ *if and only if* $\tilde{\mathrm{P}}$ *does have a* $\tau$-*quantile of* $p$-*average type* $\alpha$.

The following theorem shows that for distributions having a quantile of $p$-average type the conditional quantile can be estimated by functions that approximately minimize the pinball risk.

**Theorem 2.5** *Let $p \in (0, \infty]$, $\tau \in (0,1)$, $\alpha > 0$ be real numbers, and $q := \frac{p}{p+1}$. Moreover, let P be a distribution on $X \times Y$ that has a $\tau$-quantile of $p$-average type $\alpha$. Then for all $f : X \to \mathbb{R}$ satisfying $\mathcal{R}_{L_\tau,\mathrm{P}}(f) - \mathcal{R}^*_{L_\tau,\mathrm{P}} \leq 2^{-\frac{p+2}{p+1}} \alpha^{\frac{2p}{p+1}}$ we have*

$$\|f - f^*_{\tau,\mathrm{P}}\|_{L_q(\mathrm{P}_X)} \leq \sqrt{2} \, \|b^{-1}\|^{1/2}_{L_p(\mathrm{P}_X)} \sqrt{\mathcal{R}_{L_\tau,\mathrm{P}}(f) - \mathcal{R}^*_{L_\tau,\mathrm{P}}} \, .$$

Our next goal is to establish an oracle inequality for SVMs defined by (1). To this end let us assume $Y = [-1, 1]$. Then we have $L_\tau(y, \bar{t}) \leq L_\tau(y, t)$ for all $y \in Y$, $t \in \mathbb{R}$, where $\bar{t}$ denotes $t$ clipped to the interval $[-1, 1]$, i.e., $\bar{t} := \max\{-1, \min\{1, t\}\}$. Since this yields $\mathcal{R}_{L_\tau,\mathrm{P}}(\bar{f}) \leq \mathcal{R}_{L_\tau,\mathrm{P}}(f)$ for all functions $f : X \to \mathbb{R}$ we will focus on clipped functions $\bar{f}$ in the following. To describe the approximation error of SVMs we need the *approximation error function* $A(\lambda) := \inf_{f \in H} \lambda \|f\|^2_H + \mathcal{R}_{L_\tau,\mathrm{P}}(f) - \mathcal{R}^*_{L_\tau,\mathrm{P}}$, $\lambda > 0$. Recall that [8] showed $\lim_{\lambda \to 0} A(\lambda) = 0$ if the RKHS $H$ is dense in $L_1(\mathrm{P}_X)$. We also need the *covering numbers* which for $\varepsilon > 0$ are defined by

$$\mathcal{N}\big(B_H, \varepsilon, L_2(\mu)\big) := \min\big\{n \geq 1 : \exists\, x_1, \ldots, x_n \in L_2(\mu) \text{ with } B_H \subset \cup_{i=1}^n (x_i + \varepsilon B_{L_2(\mu)})\big\}, \quad (4)$$

where $\mu$ is a distribution on $X$, and $B_H$ and $B_{L_2(\mu)}$ denote the closed unit balls of $H$ and the Hilbert space $L_2(\mu)$, respectively. Given a finite sequence $D = ((x_1, y_1), \ldots, (x_n, y_n)) \in (X \times Y)^n$ we write $D_X := (x_1, \ldots, x_n)$, and $\mathcal{N}(B_H, \varepsilon, L_2(D_X)) := \mathcal{N}(B_H, \varepsilon, L_2(\mu))$ if $\mu$ is the empirical measure defined by $D_X$. Finally, we write $L_\tau \circ f$ for the function $(x, y) \mapsto L_\tau(y, f(x))$. With these preparations we can now recall the following oracle inequality shown in more generality in [9].

**Theorem 2.6** *Let P be a distribution on $X \times [-1, 1]$ for which there exist constants $v \geq 1$, $\vartheta \in [0, 1]$ with*

$$\mathbb{E}_{\mathrm{P}}\big(L_\tau \circ \bar{f} - L_\tau \circ f^*_{\tau,\mathrm{P}}\big)^2 \leq v \big(\mathbb{E}_{\mathrm{P}}(L_\tau \circ \bar{f} - L_\tau \circ f^*_{\tau,\mathrm{P}})\big)^\vartheta \quad (5)$$

*for all $f : X \to \mathbb{R}$. Moreover, let $H$ be a RKHS over $X$ for which there exist $\varrho \in (0, 1)$ and $a \geq 1$ with*

$$\sup_{D \in (X \times Y)^n} \log \mathcal{N}\big(B_H, \varepsilon, L_2(D_X)\big) \leq a\varepsilon^{-2\varrho}, \qquad \varepsilon > 0. \quad (6)$$

*Then there exists a constant $K_{\varrho,v}$ depending only on $\varrho$ and $v$ such that for all $\varsigma \geq 1$, $n \geq 1$, and $\lambda > 0$ we have with probability not less than $1 - 3e^{-\varsigma}$ that*

$$\mathcal{R}_{L_\tau,\mathrm{P}}(\bar{f}_{D,\lambda}) - \mathcal{R}^*_{L_\tau,\mathrm{P}} \leq 8A(\lambda) + 30\sqrt{\frac{A(\lambda)}{\lambda}\frac{\varsigma}{n}} + \left(\frac{K_{\varrho,v}a}{\lambda^\varrho n}\right)^{\frac{1}{2-\vartheta+\varrho(\vartheta-1)}} + \frac{K_{\varrho,v}a}{\lambda^\varrho n} + 5\left(\frac{32v\varsigma}{n}\right)^{\frac{1}{2-\vartheta}}.$$

Moreover, [9] showed that oracle inequalities of the above type can be used to establish learning rates and to investigate data-dependent parameter selection strategies. For example if we assume that there exist constants $c > 0$ and $\beta \in (0, 1]$ such that $A(\lambda) \leq c\lambda^\beta$ for all $\lambda > 0$ then $\mathcal{R}_{L_\tau,\mathrm{P}}(\bar{f}_{T,\lambda_n})$ converges to $\mathcal{R}^*_{L_\tau,\mathrm{P}}$ with rate $n^{-\gamma}$ where $\gamma := \min\{\frac{\beta}{\beta(2-\vartheta+\varrho(\vartheta-1))+\varrho}, \frac{2\beta}{\beta+1}\}$ and $\lambda_n = n^{-\gamma/\beta}$. Moreover, [9] shows that this rate can also be achieved by selecting $\lambda$ in a data-dependent way with the help of a validation set. Let us now consider how these learning rates in terms of risks translate into rates for $\|\bar{f}_{T,\lambda} - f^*_{\tau,\mathrm{P}}\|_{L_q(\mathrm{P}_X)}$. To this end we assume that P has a $\tau$-quantile of $p$-average type $\alpha$ for $\tau \in (0, 1)$. Using the Lipschitz continuity of $L_\tau$ and Theorem 2.5 we then obtain

$$\mathbb{E}_{\mathrm{P}}\big(L_\tau \circ \bar{f} - L_\tau \circ f^*_{\tau,\mathrm{P}}\big)^2 \leq \mathbb{E}_{\mathrm{P}}|\bar{f} - f^*_{\tau,\mathrm{P}}|^2 \leq \|\bar{f} - f^*_{\tau,\mathrm{P}}\|^{2-q}_\infty \mathbb{E}_{\mathrm{P}}|\bar{f} - f^*_{\tau,\mathrm{P}}|^q \leq c\big(\mathcal{R}_{L_\tau,\mathrm{P}}(\bar{f}) - \mathcal{R}^*_{L_\tau,\mathrm{P}}\big)^{q/2}$$

for all $f$ satisfying $\mathcal{R}_{L_\tau,\mathrm{P}}(\bar{f}) - \mathcal{R}^*_{L_\tau,\mathrm{P}} \leq 2^{-\frac{p+2}{p+1}} \alpha^{\frac{2p}{p+1}}$, i.e. we have a variance bound (5) for $\vartheta := q/2$ and clipped functions with small excess risk. Arguing carefully to handle the restriction on $\bar{f}$ we then see that $\|\bar{f}_{T,\lambda} - f^*_{\tau,\mathrm{P}}\|_{L_q(\mathrm{P}_X)}$ can converge as fast as $n^{-\gamma}$, where

$$\gamma := \min\left\{\frac{\beta}{\beta(4-q+\varrho(q-2))+2\varrho}, \frac{\beta}{\beta+1}\right\}.$$

To illustrate the latter let us assume that $H$ is a Sobolev space $W^m(X)$ of order $m \in \mathbb{N}$ over $X$, where $X$ is the unit ball in $\mathbb{R}^d$. Recall from [3] that $H$ satisfies (6) for $\varrho := d/(2m)$ if $m > d/2$ and

in this case $H$ also consists of continuous functions. Furthermore, assume that we are in the ideal situation $f^*_{\tau,\mathrm{P}} \in W^m(X)$ which implies $\beta = 1$. Then the learning rate for $\|\bar{f}_{T,\lambda} - f^*_{\tau,\mathrm{P}}\|_{L_q(\mathrm{P}_X)}$ becomes $n^{-1/(4-q(1-\varrho))}$, which for $\infty$-average type distributions reduces to $n^{-2m/(6m+d)} \approx n^{-1/3}$.

Let us finally investigate whether the $\epsilon$-insensitive loss defined by $L(y,t) := \max\{0, |y-t| - \epsilon\}$ for $y, t \in \mathbb{R}$ and fixed $\epsilon > 0$, can be used to estimate the median, i.e. the $(1/2)$-quantile.

**Theorem 2.7** *Let $L$ be the $\epsilon$-insensitive loss for some $\epsilon > 0$ and $\mathrm{P}$ be a distribution on $X \times \mathbb{R}$ which has a unique median $f^*_{1/2,\mathrm{P}}$. Furthermore, assume that all conditional distributions $\mathrm{P}(\cdot|x)$, $x \in X$, are atom-free, i.e. $\mathrm{P}(\{y\}|x) = 0$ for all $y \in \mathbb{R}$, and symmetric, i.e. $\mathrm{P}(h(x)+A|x) = \mathrm{P}(h(x)-A|x)$ for all measurable $A \subset \mathbb{R}$ and a suitable function $h : X \to \mathbb{R}$. If for the conditional distributions have a positive mass concentrated around $f^*_{1/2,\mathrm{P}} \pm \epsilon$ then $f^*_{1/2,\mathrm{P}}$ is the only minimizer of $\mathcal{R}_{L,\mathrm{P}}$.*

Note that using [7] one can show that for distributions specified in the above theorem the SVM using the $\epsilon$-insensitive loss approximates $f^*_{1/2,\mathrm{P}}$ whenever the SVM is $\mathcal{R}_{L,\mathrm{P}}$-consistent, i.e. $\mathcal{R}_{L,\mathrm{P}}(f_{T,\lambda}) \to \mathcal{R}^*_{L,\mathrm{P}}$ in probability, see [2]. More advanced results in the sense of Theorem 2.5 seem also possible, but are out of the scope of this paper.

## 3   Proofs

Let us first recall some notions from [7] who investigated surrogate losses in general and the question how approximate risk minimizers approximate exact risk minimizers in particular. To this end let $L : X \times Y \times \mathbb{R} \to [0,\infty)$ be a measurable function which we call a loss in the following. For a distribution $\mathrm{P}$ and an $f : X \to \mathbb{R}$ the $L$-risk is then defined by $\mathcal{R}_{L,\mathrm{P}}(f) := \mathbb{E}_{(x,y)\sim\mathrm{P}} L(x,y,f(x))$, and, as usual, the Bayes $L$-risk, is denoted by $\mathcal{R}^*_{L,\mathrm{P}} := \inf \mathcal{R}_{L,\mathrm{P}}(f)$, where the infimum is taken over all (measurable) $f : X \to \mathbb{R}$. In addition, given a distribution $\mathrm{Q}$ on $Y$ the *inner $L$-risks* were defined by $\mathcal{C}_{L,\mathrm{Q},x}(t) := \int_Y L(x,y,t)\, d\mathrm{Q}(y)$, $x \in X$, $t \in \mathbb{R}$, and the *minimal inner $L$-risks* were denoted by $\mathcal{C}^*_{L,\mathrm{Q},x} := \inf \mathcal{C}_{L,\mathrm{Q},x}(t)$, $x \in X$, where the infimum is taken over all $t \in \mathbb{R}$. Moreover, following [7] we usually omit the indexes $x$ or $\mathrm{Q}$ if $L$ is independent of $x$ or $y$, respectively. Obviously, we have

$$\mathcal{R}_{L,\mathrm{P}}(f) = \int_X \mathcal{C}_{L,\mathrm{P}(\,\cdot\,|x),x}\big(f(x)\big)\, d\mathrm{P}_X(x)\,, \tag{7}$$

and [7, Theorem 3.2] further shows that $x \mapsto \mathcal{C}^*_{L,\mathrm{P}(\,\cdot\,|x),x}$ is measurable if the $\sigma$-algebra on $X$ is complete. In this case it was also shown that the intuitive formula $\mathcal{R}^*_{L,\mathrm{P}} = \int_X \mathcal{C}^*_{L,\mathrm{P}(\,\cdot\,|x),x}\, d\mathrm{P}_X(x)$ holds, i.e. the Bayes $L$-risk is obtained by minimizing the inner risks and subsequently integrating with respect to the marginal distribution $\mathrm{P}_X$. Based on this observation the basic idea in [7] is to consider both steps separately. In particular, it turned out that the sets of *$\varepsilon$-approximate minimizers* $\mathcal{M}_{L,\mathrm{Q},x}(\varepsilon) := \big\{t \in \mathbb{R} : \mathcal{C}_{L,\mathrm{Q},x}(t) < \mathcal{C}^*_{L,\mathrm{Q},x} + \varepsilon\big\}$, $\varepsilon \in [0,\infty]$, and the set of *exact minimizers* $\mathcal{M}_{L,\mathrm{Q},x}(0^+) := \bigcap_{\varepsilon>0} \mathcal{M}_{L,\mathrm{Q},x}(\varepsilon)$ play a crucial role. As in [7] we again omit the subscripts $x$ and $\mathrm{Q}$ in these definitions if $L$ happens to be independent of $x$ or $y$, respectively.

Now assume we have two losses $L_{\mathrm{tar}} : X \times Y \times \mathbb{R} \to [0,\infty]$ and $L_{\mathrm{sur}} : X \times Y \times \mathbb{R} \to [0,\infty]$, and that our goal is to estimate the excess $L_{\mathrm{tar}}$-risk by the excess $L_{\mathrm{sur}}$-risk. This issue was investigated in [7], where the main device was the so-called *calibration function* $\delta_{\max}(\,\cdot\,,\mathrm{Q},x)$ defined by

$$\delta_{\max}(\varepsilon,\mathrm{Q},x) := \begin{cases} \inf_{t \in \mathbb{R}\backslash\mathcal{M}_{L_{\mathrm{tar}},\mathrm{Q},x}(\varepsilon)} \mathcal{C}_{L_{\mathrm{sur}},\mathrm{Q},x}(t) - \mathcal{C}^*_{L_{\mathrm{sur}},\mathrm{Q},x} & \text{if } \mathcal{C}^*_{L_{\mathrm{sur}},\mathrm{Q},x} < \infty\,, \\ \infty & \text{if } \mathcal{C}^*_{L_{\mathrm{sur}},\mathrm{Q},x} = \infty\,, \end{cases}$$

for all $\varepsilon \in [0,\infty]$. In the following we sometimes write $\delta_{\max,L_{\mathrm{tar}},L_{\mathrm{sur}}}(\varepsilon,\mathrm{Q},x) := \delta_{\max}(\varepsilon,\mathrm{Q},x)$ whenever we need to explicitly mention the target and surrogate losses. In addition, we follow our convention which omits $x$ or $\mathrm{Q}$ whenever this is possible. Now recall that [7, Lemma 2.9] showed

$$\delta_{\max}\big(\mathcal{C}_{L_{\mathrm{tar}},\mathrm{Q},x}(t) - \mathcal{C}^*_{L_{\mathrm{tar}},\mathrm{Q},x},\mathrm{Q},x\big) \leq \mathcal{C}_{L_{\mathrm{sur}},\mathrm{Q},x}(t) - \mathcal{C}^*_{L_{\mathrm{sur}},\mathrm{Q},x}\,, \qquad t \in \mathbb{R} \tag{8}$$

if both $\mathcal{C}^*_{L_{\mathrm{tar}},\mathrm{Q},x} < \infty$ and $\mathcal{C}^*_{L_{\mathrm{sur}},\mathrm{Q},x} < \infty$. Before we use (8) to establish an inequality between the excess risks of $L_{\mathrm{tar}}$ and $L_{\mathrm{sur}}$, we finally recall that the Fenchel-Legendre bi-conjugate $g^{**} : I \to [0,\infty]$ of a function $g : I \to [0,\infty]$ defined on an interval $I$ is the largest convex function $h : I \to [0,\infty]$ satisfying $h \leq g$. In addition, we write $g^{**}(\infty) := \lim_{t\to\infty} g^{**}(t)$ if $I = [0,\infty)$. With these preparations we can now establish the following generalization of [7, Theorem 2.18].

**Theorem 3.1** *Let* $\mathrm{P}$ *be a distribution on* $X \times Y$ *with* $\mathcal{R}^*_{L_{\mathrm{tar}},\mathrm{P}} < \infty$ *and* $\mathcal{R}^*_{L_{\mathrm{sur}},\mathrm{P}} < \infty$ *and assume that there exist* $p \in (0, \infty]$ *and functions* $b : X \to [0, \infty]$ *and* $\delta : [0, \infty) \to [0, \infty)$ *such that*

$$\delta_{\max}(\varepsilon, \mathrm{P}(\,\cdot\,|x), x) \geq b(x)\,\delta(\varepsilon), \qquad \varepsilon \geq 0, \ x \in X, \tag{9}$$

*and* $b^{-1} \in L_p(\mathrm{P}_X)$. *Then for* $q := \frac{p}{p+1}$, $\bar{\delta} := \delta^q : [0, \infty) \to [0, \infty)$, *and all* $f : X \to \mathbb{R}$ *we have*

$$\bar{\delta}^{**}\big(\mathcal{R}_{L_{\mathrm{tar}},\mathrm{P}}(f) - \mathcal{R}^*_{L_{\mathrm{tar}},\mathrm{P}}\big) \ \leq \ \|b^{-1}\|^q_{L_p(\mathrm{P}_X)}\big(\mathcal{R}_{L_{\mathrm{sur}},\mathrm{P}}(f) - \mathcal{R}^*_{L_{\mathrm{sur}},\mathrm{P}}\big)^q.$$

***Proof:*** Let us first consider the case $\mathcal{R}_{L_{\mathrm{tar}},\mathrm{P}}(f) < \infty$. Since $\bar{\delta}^{**}$ is convex and satisfies $\bar{\delta}^{**}(\varepsilon) \leq \bar{\delta}(\varepsilon)$ for all $\varepsilon \in [0, \infty)$ we see by Jensen's inequality that

$$\bar{\delta}^{**}\big(\mathcal{R}_{L_{\mathrm{tar}},\mathrm{P}}(f) - \mathcal{R}^*_{L_{\mathrm{tar}},\mathrm{P}}\big) \ \leq \ \int_X \bar{\delta}\big(\mathcal{C}_{L_{\mathrm{tar}},\mathrm{P}(\,\cdot\,|x),x}(t) - \mathcal{C}^*_{L_{\mathrm{tar}},\mathrm{P}(\,\cdot\,|x),x}\big)\,d\mathrm{P}_X(x) \tag{10}$$

Moreover, using (8) and (9) we obtain

$$b(x)\,\delta\big(\mathcal{C}_{L_{\mathrm{tar}},\mathrm{P}(\,\cdot\,|x),x}(t) - \mathcal{C}^*_{L_{\mathrm{tar}},\mathrm{P}(\,\cdot\,|x),x}\big) \ \leq \ \mathcal{C}_{L_{\mathrm{sur}},\mathrm{P}(\,\cdot\,|x),x}(t) - \mathcal{C}^*_{L_{\mathrm{sur}},\mathrm{P}(\,\cdot\,|x),x}$$

for $\mathrm{P}_X$-almost all $x \in X$ and all $t \in \mathbb{R}$. By (10), the definition of $\bar{\delta}$, and Hölder's inequality in the form of $\|\cdot\|_q \leq \|\cdot\|_p \cdot \|\cdot\|_1$, we thus find that $\bar{\delta}^{**}\big(\mathcal{R}_{L_{\mathrm{tar}},\mathrm{P}}(f) - \mathcal{R}^*_{L_{\mathrm{tar}},\mathrm{P}}\big)$ is less than or equal to

$$\left(\int_X \big(b(x)\big)^{-q}\big(\mathcal{C}_{L_{\mathrm{sur}},\mathrm{P}(\,\cdot\,|x),x}\big(f(x)\big) - \mathcal{C}^*_{L_{\mathrm{sur}},\mathrm{P}(\,\cdot\,|x),x}\big)^q\,d\mathrm{P}_X(x)\right)^{q/q}$$

$$\leq \ \left(\int_X b^{-p}\,d\mathrm{P}_X\right)^{q/p}\left(\int_X \big(\mathcal{C}_{L_{\mathrm{sur}},\mathrm{P}(\,\cdot\,|x),x}\big(f(x)\big) - \mathcal{C}^*_{L_{\mathrm{sur}},\mathrm{P}(\,\cdot\,|x),x}\big)\,d\mathrm{P}_X(x)\right)^q$$

$$\leq \ \|b^{-1}\|^q_{L_p(\mathrm{P}_X)}\big(\mathcal{R}_{L_{\mathrm{sur}},\mathrm{P}}(f) - \mathcal{R}^*_{L_{\mathrm{tar}},\mathrm{P}}\big)^q.$$

Let us finally deal with the case $\mathcal{R}_{L_{\mathrm{tar}},\mathrm{P}}(f) = \infty$. If $\bar{\delta}^{**}(\infty) = 0$ there is nothing to prove and hence we assume $\bar{\delta}^{**}(\infty) > 0$. Following the proof of [7, Theorem 2.13] we then see that there exist constants $c_1, c_2 \in (0, \infty)$ satisfying $t \leq c_1\bar{\delta}^{**}(t) + c_2$ for all $t \in [0, \infty]$. From this we obtain

$$\infty \ = \ \mathcal{R}_{L_{\mathrm{tar}},\mathrm{P}}(f) - \mathcal{R}^*_{L_{\mathrm{tar}},\mathrm{P}} \leq c_1 \int_X \bar{\delta}^{**}\big(\mathcal{C}_{L_{\mathrm{tar}},\mathrm{P}(\,\cdot\,|x),x}(t) - \mathcal{C}^*_{L_{\mathrm{tar}},\mathrm{P}(\,\cdot\,|x),x}\big)\,d\mathrm{P}_X(x) + c_2$$

$$\leq \ c_1 \int_X \big(b(x)\big)^{-q}\big(\mathcal{C}_{L_{\mathrm{sur}},\mathrm{P}(\,\cdot\,|x),x}\big(f(x)\big) - \mathcal{C}^*_{L_{\mathrm{sur}},\mathrm{P}(\,\cdot\,|x),x}\big)^q\,d\mathrm{P}_X(x) + c_2\,,$$

where the last step is analogous to our considerations for $\mathcal{R}_{L_{\mathrm{tar}},\mathrm{P}}(f) < \infty$. By $b^{-1} \in L_p(\mathrm{P}_X)$ and Hölder's inequality we then conclude $\mathcal{R}_{L_{\mathrm{sur}},\mathrm{P}}(f) - \mathcal{R}^*_{L_{\mathrm{sur}},\mathrm{P}} = \infty$. ∎

Our next goal is to determine the inner risks and their minimizers for the pinball loss. To this end recall (see, e.g., [1, Theorem 23.8]) that given a distribution $\mathrm{Q}$ on $\mathbb{R}$ and a *non-negative* function $g : X \to [0, \infty)$ we have

$$\int_{\mathbb{R}} g\,d\mathrm{Q} = \int_0^\infty \mathrm{Q}(g \geq s)\,ds\,. \tag{11}$$

**Proposition 3.2** *Let* $\tau \in (0, 1)$ *and* $\mathrm{Q}$ *be a distribution on* $\mathbb{R}$ *with* $\mathcal{C}^*_{L_\tau,\mathrm{Q}} < \infty$ *and* $t^*$ *be a* $\tau$-*quantile of* $\mathrm{Q}$. *Then there exist* $q_+, q_- \in [0, \infty)$ *with* $q_+ + q_- = \mathrm{Q}(\{t^*\})$, *and for all* $t \geq 0$ *we have*

$$\mathcal{C}_{L_\tau,\mathrm{Q}}(t^* + t) - \mathcal{C}_{L_\tau,\mathrm{Q}}(t^*) \ = \ tq_+ + \int_0^t \mathrm{Q}\big((t^*, t^* + s)\big)\,ds\,, \quad \textit{and} \tag{12}$$

$$\mathcal{C}_{L_\tau,\mathrm{Q}}(t^* - t) - \mathcal{C}_{L_\tau,\mathrm{Q}}(t^*) \ = \ tq_- + \int_0^t \mathrm{Q}\big((t^* - s, t^*)\big)\,ds\,. \tag{13}$$

***Proof:*** Let us consider the distribution $\mathrm{Q}^{(t^*)}$ defined by $\mathrm{Q}^{(t^*)}(A) := \mathrm{Q}(t^* + A)$ for all measurable $A \subset \mathbb{R}$. Then it is not hard to see that $0$ is a $\tau$-quantile of $\mathrm{Q}^{(t^*)}$. Moreover, we obviously have $\mathcal{C}_{L_\tau,\mathrm{Q}}(t^* + t) = \mathcal{C}_{L_\tau,\mathrm{Q}^{(t^*)}}(t)$ and hence we may assume without loss of generality that $t^* = 0$. Then our assumptions together with $\mathrm{Q}((-\infty, 0]) + \mathrm{Q}([0, \infty)) = 1 + \mathrm{Q}(\{0\})$ yield $\tau \leq \mathrm{Q}((-\infty, 0]) \leq \tau + \mathrm{Q}(\{0\})$, i.e., there exists a $q_+$ satisfying $0 \leq q_+ \leq \mathrm{Q}(\{0\})$ and

$$\mathrm{Q}((-\infty, 0]) = \tau + q_+\,. \tag{14}$$

Let us now compute the inner risks of $L_\tau$. To this end we first assume $t \geq 0$. Then we have

$$\int_{y<t} (y-t)\,d\mathrm{Q}(y) = \int_{y<0} y\,d\mathrm{Q}(y) - t\mathrm{Q}((-\infty,t)) + \int_{0\leq y<t} y\,d\mathrm{Q}(y)$$

and $\int_{y\geq t}(y-t)\,d\mathrm{Q}(y) = \int_{y\geq 0} y\,d\mathrm{Q}(y) - t\mathrm{Q}([t,\infty)) - \int_{0\leq y<t} y\,d\mathrm{Q}(y)$ and hence we obtain

$$
\begin{aligned}
\mathcal{C}_{L_\tau,\mathrm{Q}}(t) &= (\tau-1)\int_{y<t}(y-t)\,d\mathrm{Q}(y) + \tau\int_{y\geq t}(y-t)\,d\mathrm{Q}(y) \\
&= \mathcal{C}_{L_\tau,\mathrm{Q}}(0) - \tau t + t\mathrm{Q}((-\infty,0)) + t\mathrm{Q}([0,t)) - \int_{0\leq y<t} y\,d\mathrm{Q}(y)\,.
\end{aligned}
$$

Moreover, using (11) we find

$$t\mathrm{Q}([0,t)) - \int_{0\leq y<t} y\,d\mathrm{Q}(y) = \int_0^t \mathrm{Q}([0,t))ds - \int_0^t \mathrm{Q}([s,t))\,ds = t\mathrm{Q}(\{0\}) + \int_0^t \mathrm{Q}((0,s))ds\,,$$

and since (14) implies $\mathrm{Q}((-\infty,0)) + \mathrm{Q}(\{0\}) = \mathrm{Q}((-\infty,0]) = \tau + q_+$ we thus obtain (12). Now (13) can be derived from (12) by considering the pinball loss with parameter $1-\tau$ and the distribution $\bar{\mathrm{Q}}$ defined by $\bar{\mathrm{Q}}(A) := \mathrm{Q}(-A)$, $A \subset \mathbb{R}$ measurable. This further yields a $q_-$ satisfying $0 \leq q_- \leq \mathrm{Q}(\{0\})$ and $\bar{\mathrm{Q}}([0,\infty)) = 1 - \tau + q_-$. By (14) we then find $q_+ + q_- = \mathrm{Q}(\{0\})$. ∎

For the proof of Theorem 2.5 we recall a few more concepts from [7]. To this end let us now assume that our loss is independent of $x$, i.e. we consider a measurable function $L : Y \times \mathbb{R} \to [0,\infty]$. We write

$$\mathcal{Q}_{\min}(L) := \left\{ \mathrm{Q} \in \mathcal{Q}_{\min}(L) : \exists t^*_{L,\mathrm{Q}} \in \mathbb{R} \text{ such that } \mathcal{M}_{L,\mathrm{Q}}(0^+) = \{t^*_{L,\mathrm{Q}}\} \right\},$$

i.e. $\mathcal{Q}_{\min}(L)$ contains the distributions on $Y$ whose inner $L$-risks have exactly one exact minimizer. Furthermore, note that this definition immediately yields $\mathcal{C}^*_{L,\mathrm{Q}} < \infty$ for all $\mathrm{Q} \in \mathcal{Q}_{\min}(L)$. Following [7] we now define the *self-calibration loss* of $L$ by

$$\check{L}(\mathrm{Q},t) := |t - t^*_{L,\mathrm{Q}}|, \qquad \mathrm{Q} \in \mathcal{Q}_{\min}(L),\ t \in \mathbb{R}\,. \tag{15}$$

This loss is a so-called template loss in the sense of [7], i.e., for a given distribution $\mathrm{P}$ on $X \times Y$, where $X$ has a complete $\sigma$-algebra and $\mathrm{P}(\cdot\,|x) \in \mathcal{Q}_{\min}(L)$ for $\mathrm{P}_X$-almost all $x \in X$, the $\mathrm{P}$-*instance* $\check{L}_\mathrm{P}(x,t) := |t - t^*_{L,\mathrm{P}(\cdot\,|x)}|$ is measurable and hence a loss. [7] extended the definition of inner risks to the self-calibration loss by setting $\mathcal{C}_{\check{L},\mathrm{Q}}(t) := \check{L}(\mathrm{Q},t)$, and based on this the minimal inner risks and their (approximate) minimizers were defined in the obvious way. Moreover, the *self-calibration function* was defined by $\delta_{\max,\check{L},L}(\varepsilon,\mathrm{Q}) = \inf_{t\in\mathbb{R};\,|t-t^*_{L,\mathrm{Q}}|\geq\varepsilon} \mathcal{C}_{L,\mathrm{Q}}(t) - \mathcal{C}^*_{L,\mathrm{Q}}$. As shown in [7] this self-calibration function has two important properties: first it satisfies

$$\delta_{\max,\check{L},L}\left(|t-t^*_{L,\mathrm{Q}}|,\mathrm{Q}\right) \leq \mathcal{C}_{L,\mathrm{Q}}(t) - \mathcal{C}^*_{L,\mathrm{Q}}\,, \qquad t \in \mathbb{R}, \tag{16}$$

i.e. it measures how well approximate $L$-risk minimizers $t$ approximate the true minimizer $t^*_{L,\mathrm{Q}}$, and second it equals the calibration function of the $\mathrm{P}$-instance $\check{L}_\mathrm{P}$, i.e.

$$\delta_{\max,\check{L}_\mathrm{P},L}(\varepsilon,\mathrm{P}(\cdot\,|x),x) = \delta_{\max,\check{L},L}(\varepsilon,\mathrm{P}(\cdot\,|x))\,, \qquad \varepsilon \in [0,\infty],\ x \in X. \tag{17}$$

In other words, the self-calibration function can be utilized in Theorem 3.1.

***Proof of Theorem 2.5:*** Let $\mathrm{Q}$ be a distribution on $\mathbb{R}$ with $\mathcal{C}^*_{L,\mathrm{Q}} < \infty$ and $t^*$ be the *only* $\tau$-quantile of $\mathrm{Q}$. Then the formulas of Proposition 3.2 show

$$\delta_{\max,\check{L},L}(\varepsilon,\mathrm{Q}) = \min\left\{ \varepsilon q_+ + \int_0^\varepsilon \mathrm{Q}\big((t^*,t^*+s)\big)\,ds,\ \varepsilon q_- + \int_0^\varepsilon \mathrm{Q}\big((t^*-s,t^*)\big)\,ds \right\}, \quad \varepsilon \geq 0,$$

where $q_+$ and $q_-$ are the real numbers defined in Proposition 3.2. Let us additionally assume that the $\tau$-quantile $t^*$ is of type $\alpha$. For the Huber type function $\delta(\varepsilon) := \varepsilon^2/2$ if $\varepsilon \in [0,\alpha]$, and $\delta(\varepsilon) := \alpha\varepsilon - \alpha^2/2$ if $\varepsilon > \alpha$, a simple calculation then yields $\delta_{\max,\check{L},L}(\varepsilon,\mathrm{Q}) \geq c_\mathrm{Q}\delta(\varepsilon)$, where $c_\mathrm{Q}$ is the constant satisfying (3). Let us further define $\bar{\delta} : [0,\infty) \to [0,\infty)$ by $\bar{\delta}(\varepsilon) := \delta^q(\varepsilon^{1/q})$, $\varepsilon \geq 0$. In view of Theorem 3.1 we then need to find a convex function $\hat{\delta} : [0,\infty) \to [0,\infty)$ such that $\hat{\delta} \leq \bar{\delta}$. To this end we define $\hat{\delta}(\varepsilon) := s_p^p \varepsilon^2$ if $\varepsilon \in \left[0, s_p a_p\right]$ and $\hat{\delta}(\varepsilon) := a_p\big(\varepsilon - s_p^{p+2}a_p\big)$ if $\varepsilon > s_p a_p$, where $a_p := \alpha^q$ and $s_p := 2^{-q/p}$. Then $\hat{\delta} : [0,\infty) \to [0,\infty)$ is continuously differentiable and its derivative is increasing, and thus $\hat{\delta}$ is convex. Moreover, we have $\hat{\delta}' \leq \bar{\delta}'$ and hence $\hat{\delta} \leq \bar{\delta}$ which in turn implies $\hat{\delta} \leq \bar{\delta}^{**}$. Now we find the assertion by (16), (17), and Theorem 3.1. ∎

The proof of Theorem 2.7 follows immediately from the following lemma.

**Lemma 3.3** *Let $Q$ be a symmetric, atom-free distribution on $\mathbb{R}$ with median $t^* = 0$. Then for $\epsilon > 0$ and $L$ being the $\epsilon$-insensitive loss we have $\mathcal{C}_{L,Q}(0) = \mathcal{C}^*_{L,Q} = 2\int_\epsilon^\infty Q[s,\infty)ds$ and if $\mathcal{C}_{L,Q}(0) < \infty$ we further have*

$$\mathcal{C}_{L,Q}(t) - \mathcal{C}_{L,Q}(0) = \int_{\epsilon-t}^\epsilon Q[s,\epsilon]\,ds + \int_\epsilon^{\epsilon+t} Q[\epsilon,s]\,ds, \qquad\qquad \text{if } t \in [0,\epsilon],$$

$$\mathcal{C}_{L,Q}(t) - \mathcal{C}_{L,Q}(\epsilon) = \int_0^{t-\epsilon} Q[s,\infty)\,ds - \int_{2\epsilon}^{\epsilon+t} Q[s,\infty)\,ds + 2\int_0^{t-\epsilon} Q[0,s]\,ds \geq 0, \quad \text{if } t > \epsilon.$$

*In particular, if $Q[\epsilon - \delta, \epsilon + \delta] = 0$ for some $\delta > 0$ then $\mathcal{C}_{L,Q}(\delta) = \mathcal{C}^*_{L,Q}$.*

**Proof:** Because $L(y,t) = L(-y,-t)$ for all $y, t \in \mathbb{R}$ we only have to consider $t \geq 0$. For later use we note that for $0 \leq a \leq b \leq \infty$ Equation (11) yields

$$\int_a^b y\, dQ(y) = aQ([a,b]) + \int_a^b Q([s,b])ds. \tag{18}$$

Moreover, the definition of $L$ implies

$$\mathcal{C}_{L,Q}(t) = \int_{-\infty}^{t-\epsilon} t - y - \epsilon\, dQ(y) + \int_{t+\epsilon}^\infty y - \epsilon - t\, dQ(y).$$

Using the symmetry of $Q$ yields $-\int_{-\infty}^{t-\epsilon} y\, dQ(y) = \int_{\epsilon-t}^\infty y\, dQ(y)$ and hence we obtain

$$\mathcal{C}_{L,Q}(t) = \int_0^{t-\epsilon} Q(-\infty, t-\epsilon]ds - \int_0^{t+\epsilon} Q[t+\epsilon,\infty)ds + \int_{\epsilon-t}^{t+\epsilon} y\, dQ(y) + 2\int_{t+\epsilon}^\infty y\, dQ(y). \tag{19}$$

Let us first consider the case $t \geq \epsilon$. Then the symmetry of $Q$ yields $\int_{\epsilon-t}^{t+\epsilon} y\, dQ(y) = \int_{t-\epsilon}^{t+\epsilon} y\, dQ(y)$, and hence (18) implies

$$\mathcal{C}_{L,Q}(t) = \int_0^{t-\epsilon} Q[\epsilon - t, \infty)ds + \int_0^{t-\epsilon} Q[t-\epsilon, t+\epsilon]\, ds + \int_{t-\epsilon}^{t+\epsilon} Q[s, t+\epsilon]\, ds$$

$$+ 2\int_{t+\epsilon}^\infty Q[s,\infty)\, ds + \int_0^{t+\epsilon} Q[t+\epsilon,\infty)\, ds.$$

Using

$$\int_{t-\epsilon}^{t+\epsilon} Q[s, t+\epsilon)\, ds = \int_0^{t+\epsilon} Q[s, t+\epsilon)\, ds - \int_0^{t-\epsilon} Q[s, t+\epsilon)\, ds$$

we further obtain

$$\int_{t-\epsilon}^{t+\epsilon} Q[s, t+\epsilon)\, ds + \int_0^{t+\epsilon} Q[t+\epsilon,\infty)\, ds + \int_{t+\epsilon}^\infty Q[s,\infty)\, ds = \int_0^\infty Q[s,\infty)\, ds - \int_0^{t-\epsilon} Q[s, t+\epsilon)\, ds.$$

From this and $\int_0^{t-\epsilon} Q[t-\epsilon, t+\epsilon]\, ds - \int_0^{t-\epsilon} Q[s, t+\epsilon]\, ds = -\int_0^{t-\epsilon} Q[s, t-\epsilon]\, ds$ follows

$$\mathcal{C}_{L,Q}(t) = -\int_0^{t-\epsilon} Q[s, t-\epsilon]\, ds + \int_0^{t-\epsilon} Q[\epsilon - t, \infty)\, ds + \int_{t+\epsilon}^\infty Q[s,\infty)\, ds + \int_0^\infty Q[s,\infty)\, ds.$$

The symmetry of $Q$ implies $\int_0^{t-\epsilon} Q[\epsilon - t, t - \epsilon]\, ds = 2\int_0^{t-\epsilon} Q[0, t - \epsilon]\, ds$, and we get

$$-\int_0^{t-\epsilon} Q[s, t-\epsilon]\, ds + \int_0^{t-\epsilon} Q[\epsilon - t, \infty)\, ds = 2\int_0^{t-\epsilon} Q[0, s)\, ds + \int_0^{t-\epsilon} Q[s,\infty)\, ds.$$

This and

$$\int_{t+\epsilon}^\infty Q[s,\infty)\, ds + \int_0^\infty Q[s,\infty)\, ds = 2\int_{t+\epsilon}^\infty Q[s,\infty)\, ds + \int_0^{t+\epsilon} Q[s,\infty)\, ds$$

yields

$$\mathcal{C}_{L,\mathrm{Q}}(t) = 2\int_0^{t-\epsilon} \mathrm{Q}[0,s)\,ds + \int_0^{t-\epsilon} \mathrm{Q}[s,\infty)\,ds + 2\int_{t+\epsilon}^{\infty} \mathrm{Q}[s,\infty)\,ds + \int_0^{t+\epsilon} \mathrm{Q}[s,\infty)\,ds\,.$$

By

$$\int_0^{t-\epsilon} \mathrm{Q}[s,\infty)\,ds + \int_0^{t+\epsilon} \mathrm{Q}[s,\infty)\,ds = 2\int_0^{t-\epsilon} \mathrm{Q}[s,\infty)\,ds + \int_{t-\epsilon}^{t+\epsilon} \mathrm{Q}[s,\infty)\,ds$$

we obtain

$$\mathcal{C}_{L,\mathrm{Q}}(t) = 2\int_0^{t-\epsilon} \mathrm{Q}[0,\infty)\,ds + 2\int_{t+\epsilon}^{\infty} \mathrm{Q}[s,\infty)\,ds + \int_{t-\epsilon}^{t+\epsilon} \mathrm{Q}[s,\infty)\,ds$$

if $t \geq \epsilon$. Let us now consider the case $t \in [0,\epsilon]$. Analogously we obtain from (19) that

$$
\begin{aligned}
\mathcal{C}_{L,\mathrm{Q}}(t) &= \int_0^{\epsilon-t} \mathrm{Q}[\epsilon-t, t+\epsilon]\,ds + \int_{\epsilon-t}^{\epsilon+t} \mathrm{Q}[s, t+\epsilon]\,ds + 2\int_{\epsilon+t}^{\infty} \mathrm{Q}[s,\infty)\,ds \\
&\quad + 2\int_0^{\epsilon+t} \mathrm{Q}[\epsilon+t,\infty)\,ds - \int_0^{\epsilon-t} \mathrm{Q}[\epsilon-t,\infty)\,ds - \int_0^{\epsilon+t} \mathrm{Q}[\epsilon+t,\infty)\,ds\,.
\end{aligned}
$$

Combining this with

$$\int_0^{\epsilon-t} \mathrm{Q}[\epsilon-t, t+\epsilon]\,ds - \int_0^{\epsilon-t} \mathrm{Q}[\epsilon-t,\infty)\,ds = -\int_0^{\epsilon-t} \mathrm{Q}[\epsilon+t,\infty)\,ds$$

and $\int_0^{\epsilon+t} \mathrm{Q}[\epsilon+t,\infty)\,ds - \int_0^{\epsilon-t} \mathrm{Q}[\epsilon+t,\infty)\,ds = \int_{\epsilon-t}^{\epsilon+t} \mathrm{Q}[\epsilon+t,\infty)\,ds$ we get

$$
\begin{aligned}
\mathcal{C}_{L,\mathrm{Q}}(t) &= \int_{\epsilon-t}^{\epsilon+t} \mathrm{Q}[\epsilon+t,\infty)\,ds + \int_{\epsilon-t}^{\epsilon+t} \mathrm{Q}[s, t+\epsilon]\,ds + 2\int_{\epsilon+t}^{\infty} \mathrm{Q}[s,\infty)\,ds \\
&= \int_{\epsilon-t}^{\epsilon+t} \mathrm{Q}[s,\infty)\,ds + 2\int_{\epsilon+t}^{\infty} \mathrm{Q}[s,\infty)\,ds = \int_{\epsilon-t}^{\infty} \mathrm{Q}[s,\infty)\,ds + \int_{\epsilon+t}^{\infty} \mathrm{Q}[s,\infty)\,ds\,.
\end{aligned}
$$

Hence $\mathcal{C}_{L,\mathrm{Q}}(0) = 2\int_\epsilon^\infty \mathrm{Q}[s,\infty)\,ds$. The expressions for $\mathcal{C}_{L,\mathrm{Q}}(t) - \mathcal{C}_{L,\mathrm{Q}}(0)$, $t \in (0,\epsilon]$, and $\mathcal{C}_{L,\mathrm{Q}}(t) - \mathcal{C}_{L,\mathrm{Q}}(\epsilon)$, $t > \epsilon$, given in Lemma 3.3 follow by using the same arguments. Hence one exact minimizer of $\mathcal{C}_{L,\mathrm{Q}}(\cdot)$ is the median $t^* = 0$. The last assertion is a direct consequence of the formula for $\mathcal{C}_{L,\mathrm{Q}}(t) - \mathcal{C}_{L,\mathrm{Q}}(0)$ in the case $t \in (0,\epsilon]$. ∎

## References

[1] H. Bauer. *Measure and Integration Theory*. De Gruyter, Berlin, 2001.

[2] A. Christmann and I. Steinwart. Consistency and robustness of kernel based regression. *Bernoulli*, 15:799–819, 2007.

[3] D.E. Edmunds and H. Triebel. *Function Spaces, Entropy Numbers, Differential Operators*. Cambridge University Press, 1996.

[4] C. Hwang and J. Shim. A simple quantile regression via support vector machine. In *Advances in Natural Computation: First International Conference (ICNC)*, pages 512 –520. Springer, 2005.

[5] R. Koenker. *Quantile Regression*. Cambridge University Press, 2005.

[6] B. Schölkopf, A. J. Smola, R. C. Williamson, and P. L. Bartlett. New support vector algorithms. *Neural Computation*, 12:1207–1245, 2000.

[7] I. Steinwart. How to compare different loss functions. *Constr. Approx.*, 26:225–287, 2007.

[8] I. Steinwart, D. Hush, and C. Scovel. Function classes that approximate the Bayes risk. In *Proceedings of the 19th Annual Conference on Learning Theory, COLT 2006*, pages 79–93. Springer, 2006.

[9] I. Steinwart, D. Hush, and C. Scovel. An oracle inequality for clipped regularized risk minimizers. In *Advances in Neural Information Processing Systems 19*, pages 1321–1328, 2007.

[10] I. Takeuchi, Q.V. Le, T.D. Sears, and A.J. Smola. Nonparametric quantile estimation. *J. Mach. Learn. Res.*, 7:1231–1264, 2006.
